# A blind deconvolution method for neural spike identification

**Chaitanya Ekanadham**
Courant Institute
New York University
New York, NY 10012
`chaitu@math.nyu.edu`

**Daniel Tranchina**
Courant Institute
New York University
New York, NY 10012

**Eero P. Simoncelli**
Courant Institute
Center for Neural Science
Howard Hughes Medical Institute
New York University
New York, NY 10012

## Abstract

We consider the problem of estimating neural spikes from extracellular voltage recordings. Most current methods are based on clustering, which requires substantial human supervision and systematically mishandles temporally overlapping spikes. We formulate the problem as one of statistical inference, in which the recorded voltage is a noisy sum of the spike trains of each neuron convolved with its associated spike waveform. Joint maximum-a-posteriori (MAP) estimation of the waveforms and spikes is then a blind deconvolution problem in which the coefficients are sparse. We develop a block-coordinate descent procedure to approximate the MAP solution, based on our recently developed *continuous basis pursuit* method. We validate our method on simulated data as well as real data for which ground truth is available via simultaneous intracellular recordings. In both cases, our method substantially reduces the number of missed spikes and false positives when compared to a standard clustering algorithm, primarily by recovering overlapping spikes. The method offers a fully automated alternative to clustering methods that is less susceptible to systematic errors.

## 1 Introduction

The identification of individual spikes in extracellularly recorded voltage traces is a critical step in the analysis of neural data for much of systems neuroscience. One or more electrodes are embedded in neural tissue, and the voltage(s) are recorded as a function of time, with the intention of recovering the spiking activity of one or more nearby cells. Each spike appears with a stereotyped waveform, whose shape depends on the cell morphology, the filtering properties of the medium and the electrode, and the cell's position relative to the electrode. The "spike sorting" problem is that of identifying distinct cells and their respective spike times. This is a difficult statistical inverse problem, since one typically does not know the number of cells, the shapes of their waveforms, or the frequency or temporal dynamics of their spike trains (see [1] for a review).

The observed voltage is well-described as a linear superposition of the spike waveforms [1, 2, 3, 4], and thus, the problem bears resemblance to the classic sparse decomposition problem in signal processing and machine learning, where the neural waveforms are the "features" and the spike trains are the "coefficients", with the additional constraint that the features are unknown but convolutional, and the coefficients are mostly zero except for a few that are close to one. This sparse blind deconvolution problem arises in a variety of contexts other than spike sorting, including radar [5], seismology [6], and acoustic processing [7, 8].

Most current approaches to spike sorting (with notable exceptions [9, 10]) can be summarized in three steps ([1, 2]): (1) identify segments of neural activity (e.g., by thresholding the voltage), (2)

determine a low-dimensional feature representation for these segments (e.g., PCA), (3) cluster the segments in the feature space (e.g., $k$-means, mixture of Gaussians). Fig. 1 illustrates a simple version of this procedure. Segments within the same cluster are interpreted as spikes of a single neuron, whose waveform is estimated by the cluster centroid. This method works well in identifying temporally isolated spikes whose waveforms are easily distinguishable from background noise and each other. However it generally fails for segments containing more than one spike (either from the same or different neurons), because these segments do not lie close to the clusters of any individual cell [1]. This is illustrated in Figs. 1(b) 1(c), and 1(d). Several state-of-the-art methods improve or combine upon one or more of these steps (e.g., [11, 12]), but remain susceptible to these errors because they still rely on clustering. These errors are systematic, and can have important scientific consequences. For example, an unresolved question in neuroscience is whether the occurrence of correlated or synchronous spikes carries specialized information [13, 14]. In order to experimentally address this question, one needs to record from multiple neurons, and to accurately obtain their joint spiking activity. A method that systematically fails for synchronous spikes (e.g., by missing them altogether, or by incorrectly assigning them to another neuron) will lead to erroneous conclusions.

Although the limitations of clustering methods have been known within the neuroscience community for some time [1, 2, 15, 16], they remain ubiquitous. Practitioners have developed a wide range of manual adjustments to overcome these limitations, from adjusting the electrode position to isolate a single neuron, to manually performing the clustering for spike identification. However, previous studies have shown that there is great variability in manual sorting results [17], and that human choices for cluster parameters are often suboptimal [18]. As such, there is a need for a fully automated sorting method that avoids these errors. This need is becoming ever more urgent as the use of multi-electrode arrays increases ([19]): manual parameter selection for a multi-dimensional clustering problem becomes more difficult and time-consuming as the number of electrodes grows.

We formulate the spike sorting problem as a Bayesian estimation problem by incorporating a prior model for the spikes and assuming a linear-Gaussian model for the recording given the spikes [2, 4]. Although the generative model is simple, inferring the spike times and waveforms is challenging. We approximate the most likely spikes and waveform shapes given the recording (i.e. the maximum-a-posteriori, or MAP solution), by alternating between solving for the spike times while fixing the waveforms and vice versa. Solving for optimal spike times and amplitudes with fixed waveform shapes is itself an NP-hard problem, and we employ a novel method called continuous basis pursuit [20, 21], combined with iterative reweighting techniques, to approximate its solution. We compare our method with clustering on simulated and real data, demonstrating substantial reduction in spike identification errors (both misses and false positives), particularly when spikes overlap in the signal.

## 2   Model of voltage trace

The major deficiency of clustering is that each time segment is modeled as a noisy version of a single centered waveform rather than a noisy superposition of *multiple, time-shifted* waveforms. A simple generative model for the observed voltage trace $V(t)$ is summarized as follows:

$$V(t) = \sum_{n=1}^{N} \sum_{i=1}^{K_n} a_{ni} W_n(t - \tau_{ni}) + \eta(t) \tag{1}$$

$$\{\tau_{ni}\}_{i=1}^{K_n} \sim \text{Poisson Process}(\lambda_n) \qquad n = 1, ..., N$$

$$\{a_{ni}\}_{i=1}^{K_n} \sim \mathcal{N}(1, \epsilon_n^2) \qquad n = 1, ..., N \tag{2}$$

In words, the spikes are a Poisson processes with known rates $\{\lambda_n\}$ and amplitudes independently normally distributed about unity. The trace is the sum of convolutions of the spikes with their respective waveforms $\mathbf{W} \equiv \{W_n(t)\}_{n=1}^{N}$ along with Gaussian noise $\eta(t)$ (note: other log-concave noise distributions can be used). Here, $K_n$ is the (Poisson-distributed) number of spikes of the $n$'th waveform in the signal. Thus, the model accounts for superimposed spikes, variability in spike amplitude, as well as background noise. The model can easily be generalized to multielectrode recordings by making $V(t)$ and the $W_n(t)$'s vector-valued, but to simplify notation we assume a single electrode. Note also that since the model describes the full voltage trace, it does not require a

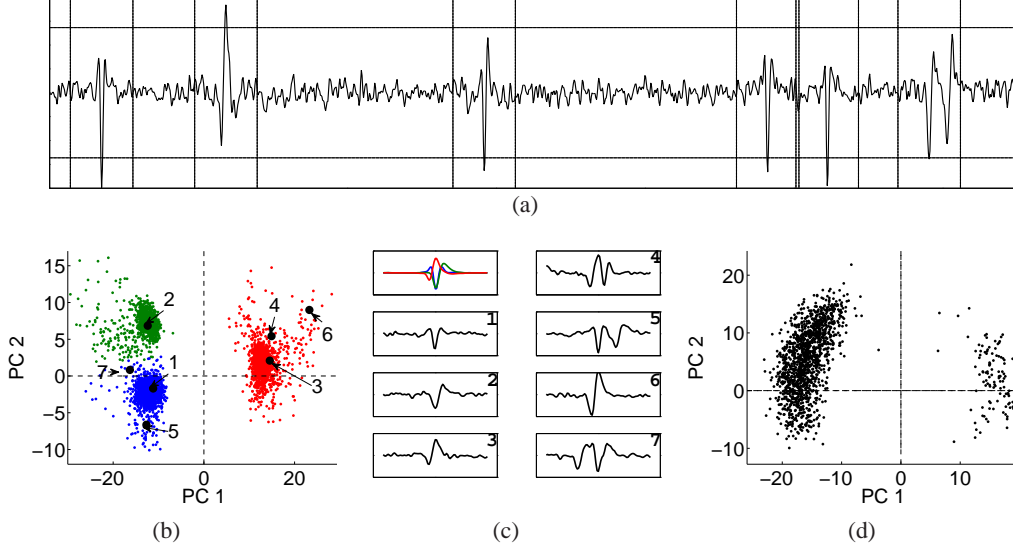

(a)

(b)                              (c)                              (d)

Figure 1: Illustration of clustering on simulated data. (a) Threshold/windowing procedure. Peaks are identified using a threshold (horizontal lines) and windows are drawn about them (vertical lines) to identify segments. (b) Plot of the segments projected onto the first two principal components. Color indicates the output of $k$-means clustering ($k = 3$). (c) The top-left plot shows the true waveforms used in this example. The other plots indicate the waveforms whose projections are the black points in (b).(d) Another example of simulated data with a single biphasic waveform (not shown). The projections of the spikes can have a non-Gaussian distribution in PC space. Two clusters arise because the waveform has two peaks around which the segments can be centered.

thresholding/windowing preprocessing stage, which can lead to additional artifacts (e.g., Fig 1(d)). The priors on the spike trains account for the observed variability in spike amplitudes and average spike rates with minimal assumptions. We are interested in the maximum-a-posteriori (MAP) solution of the waveforms and spike times and amplitudes given the observed voltage trace $V(t)$:

$$\arg\max_{\{a_{ni}\},\{\tau_{ni}\},\mathbf{W}} P(\{a_{ni}\}, \{\tau_{ni}\}, \mathbf{W}|V(t)) \tag{3}$$

$$= \arg\max_{\{a_{ni}\},\{\tau_{ni}\},\mathbf{W}} \log(P(V(t)|\{a_{ni}\}, \{\tau_{ni}\}, \mathbf{W})) + \log(P(\{a_{ni}\}, \{\tau_{ni}\}, \mathbf{W}))$$

In the following sections, we describe a procedure to approximate this solution.

## 3 Inference methods

### 3.1 Objective function

MAP estimation under the model described in Eq. (2) and Eq. (1) boils down to solving:

$$\min_{\{a_{ni}\},\{\tau_{ni}\},\mathbf{W}} \frac{1}{2}\|V(t) - \sum_{n,i} a_{ni}W_n(t-\tau_{ni})\|_{2,\Sigma}^2 + \sum_{n,i}\left[\frac{(a_{ni}-1)^2}{2\epsilon_n^2} + \frac{1}{2}\log(2\pi\epsilon_n^2) - \log(\lambda_n)\right] \tag{4}$$

where $\|\vec{x}\|_{2,\Sigma} = \|\Sigma^{-1/2}\vec{x}\|_2$ and $\Sigma$ is the noise covariance. Direct inference of the parameters is a highly nonlinear and intractable problem. However, we can make the problem tractable by using a linear representation for time-shifted waveforms. The simplest such representation uses a dictionary containing discretely time-shifted copies of the waveforms themselves $\{W_n(t - i\Delta)\}_{n,i}$. We chose

to use a more accurate and efficient dictionary to represent continuously time-shifted waveforms in the context of sparse optimization, which relies on trigonometrically varying coefficients [21]:

$$
\begin{aligned}
\sum_{n=1}^{N}\sum_{i=1}^{K_n} a_{ni} W_n(t - \tau_{ni}) &\approx \sum_{n=1}^{N}\sum_{i} \begin{pmatrix} C_n(t - i\Delta) \\ U_n(t - i\Delta) \\ V_n(t - i\Delta) \end{pmatrix}^T \begin{pmatrix} a_{ni} \\ a_{ni} r_n \cos(\frac{2\tau_{ni}\theta_n}{\delta}) \\ a_{ni} r_n \sin(\frac{2\tau_{ni}\theta_n}{\Delta}) \end{pmatrix} \\
&= \sum_{n=1}^{N}\sum_{i} \begin{pmatrix} C_n(t - i\Delta) \\ U_n(t - i\Delta) \\ V_n(t - i\Delta) \end{pmatrix}^T \begin{pmatrix} x_{ni1} \\ x_{ni2} \\ x_{ni3} \end{pmatrix} = (\mathbf{\Phi_W}\vec{x})(t)
\end{aligned}
\tag{5}
$$

The dictionary $\mathbf{\Phi_W}$ contains shifted copies of the functions $C_n(t), U_n(t), V_n(t)$ that approximate the space of time-shifted waveforms. The functions $C_n(t), U_n(t)$, and $V(t)$, as well as the constants $r_n$ and $\theta_n$ depend on the waveform $W_n(t)$ and are explained in Fig. 2(b). We can then solve the following optimization problem:

$$
\min_{\vec{x}, \mathbf{W}} F(\vec{x}, \mathbf{W}) \text{ such that } \begin{array}{l} x_{ni2} \geq r_n \cos(\theta_n) x_{ni1}, \ \forall n, i \\ \sqrt{x_{ni2}^2 + x_{ni3}^2} \leq r_n x_{ni1}, \ \forall n, i \end{array}
\tag{6}
$$

where $F(\vec{x}, \mathbf{W}) = \frac{1}{2}\|V(t) - (\mathbf{\Phi_W}\vec{x})(t)\|_{2,\Sigma}^2 - \sum_{n,i} \log\left((1 - \lambda_n\Delta)\delta(x_{ni1}) + (\lambda_n\Delta)\phi_{1,\epsilon_n^2}(x_{ni1})\right)$

where $\phi_{\mu,\sigma^2}(.)$ is the Gaussian density function. The constraints on $\vec{x}$ in Eq. (6) ensure that each triplet $(x_{ni1}, x_{ni2}, x_{ni3})$ is consistent with the mapping defined in Eq. 5, with $x_{ni1}$ being the amplitude and $\frac{\Delta}{2\theta_n}\text{atan}(x_{ni3}/x_{ni2})$ being the time-shift associated with the waveform $W_n(t)$ (see [21] for a detailed development of this approach). The constrained region, denoted by $\mathcal{C}$, is convex and is illustrated as sections of cones in Fig. 2(c). Note that we have used the Bernoulli discrete-time process with a spacing $\Delta$ (matching the interpolation dictionary spacing) to approximate the Poisson process described in Eq. (2). Even with this linear representation, the problem is not jointly convex in $\mathbf{W}$ and $\vec{x}$, and is not convex in $\vec{x}$ for fixed $\mathbf{W}$. The optimization of Eq. (6) resembles that of [22] and other sparse-coding objective functions with the following important differences: (1) the dictionary is translation-invariant and interpolates continuous time-shifts, (2) there is a constraint on the coefficients $\vec{x}$ due to the interpolation, and (3) there is a nonconvex mixture prior on the coefficients to model the spike amplitudes. We propose a block coordinate descent procedure to solve Eq. (6). After initializing $\mathbf{W}$ randomly, we iterate the following steps:

1. Given $\mathbf{W}$, approximately solve for $\vec{x}$.
2. Perform a rescaling $x_{nij} \leftarrow \frac{x_{nij}}{z_n}$ and $W_n(t) \leftarrow z_n W_n(t)$ where the $z_n$'s are chosen to optimize $F\left(\left[\frac{x_{nij}}{z_n}\right], \{z_n W_n(t)\}\right)$.
3. Given $\vec{x}$, solve for $\mathbf{W}$, constraining $\|W_n(t)\|_2$ to be less than or equal to its current value.

The first step minimizes successive convex approximations of $F$ and is the most involved of the three. The second is guaranteed to decrease $F$ and amounts to $N$ scalar optimizations. The final step minimizes the first term with respect to the waveforms while keeping the second term constant, and amounts to an $L_2$-constrained least squares problem (ridge regression) that can be solved very efficiently. The following sections provide details of each of the steps.

## 3.2 Solve spikes given waveforms

In this step we wish to minimize the function $F(\cdot, \mathbf{W})$ while ensuring that the solution lies in the convex set $\mathcal{C}$. However, this function is nonconvex and nonsmooth due to the second term in Eq. (6). This especially causes problems when the current estimates of $\mathbf{W}$ are far from the optimal values, since in this case there are many intermediate amplitudes between $0$ and $1$. To get around this, we replace each summand in the second term by a relaxation:

$$
G(x_{ni1}) = -\log\left((1 - \lambda_n\Delta)\int_0^{\infty} \frac{1}{\gamma} e^{-\frac{x_{ni1}}{\gamma}} P(\gamma) d\gamma + (\lambda_n\Delta)\phi_{1,\epsilon_n^2}(x_{ni1})\right)
\tag{7}
$$

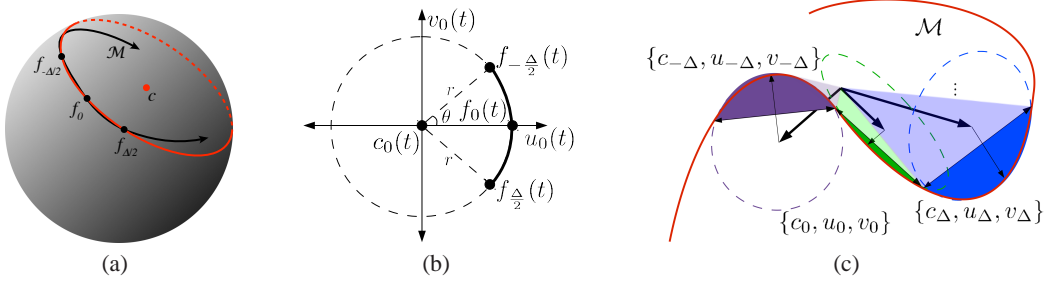

(a)             (b)             (c)

Figure 2: (a) Illustration of the circle approximation in [21]. The manifold $\mathcal{M}$ of translates of a function $f(t)$ lies on the hypersphere since translation preserves norm (black curve). This can be locally approximated by a circle (red curve). The approximation is exact at 3 equally-spaced points (black dots). (b) Visualization in the plane on which the three translates of $f(t)$ lie. The quantities $r$ and $\theta$ can be derived analytically for a fixed $f(t)$ and spacing $\Delta$. (c) These circle approximations can be linked together to form a piecewise-circular approximation of the entire manifold.

which replaces the delta function at $0$ with a mixture of exponential distributions. We chose the parameter $\gamma$ to be Gamma-distributed about a fixed small value. We solve this approximation using an iterative reweighting scheme. The weights are initialized to be uniform $w_{ni}^{(0)} = \lambda_n$, $\forall n, i$. Then the following updates are iterated computed:

$$\vec{x}^{(t+1)} \leftarrow \operatorname*{arg\,min}_{\vec{x} \in \mathcal{C}} \frac{1}{2} \| V(t) - (\mathbf{\Phi_W} \vec{x})(t) \|_2^2 + \sum_{n,i} w_{ni}^{(t)} |x_{ni1}| \tag{8}$$

$$w_{ni}^{(t+1)} \leftarrow \frac{G(x_{ni1}^{(t+1)})}{x_{ni1}^{(t+1)}} \tag{9}$$

Eq. (8) is a convex optimization that can be solved efficiently. The weights are updated so that the second term in Eq. (8) is exactly the negative log prior probability of the previous solution $\vec{x}^{(t)}$. If a coefficient is $0$, its weight is $\infty$ and the corresponding basis function is discarded. Such reweighting procedures have been used to optimize a nonconvex function by a series of convex optimizations [23, 24, 25]. Although there is no convergence guarantee, we find that it works well in practice.

### 3.3 Solve rescaling factors

The first term of $F(\vec{x}, \{W_n(t)\})$ does not change by much if one divides the coefficients $x_{nij}$ by some $z_n$ and multiplies the corresponding waveform by $z_n$ [1]. The second term does change under such a rescaling. In order to avoid the solution where the waveforms/coefficients become arbitrarily large/small, respectively, we perform a rescaling in a separate step and then optimize the waveform shapes subject to a fixed norm constraint (described in the next section). Since the second term decomposes into terms that are each only dependent on one $z_n$, we can independently solve the following scalar optimizations numerically:

$$z_n \leftarrow \operatorname*{arg\,max}_{z>0} \sum_i \log \left( (1 - \Delta \lambda_n) \frac{1}{\gamma} e^{-\frac{x_{ni1}}{z\gamma}} + \Delta \lambda_n \phi_{1, \epsilon_n^2} \left( \frac{x_{ni1}}{z} \right) \right) \qquad n = 1, ..., N \tag{10}$$

These are essentially maximum likelihood estimates of the scale factors given fixed coefficients and waveform shapes. One then performs the updates:

$$x_{nij} \leftarrow \frac{x_{nij}}{z_n} \qquad \forall n, i, j \tag{11}$$

$$W_n(t) \leftarrow z_n W_n(t) \qquad \forall n \tag{12}$$

This step is guaranteed not to increase the objective in Eq. (6) since the first term is held constant (up to a small error term, see footnote) and the second term cannot increase.

### 3.4   Solve waveforms given spikes

Given a set of coefficients $\vec{x}$, we can optimize waveform shapes by solving:

$$\min_{\mathbf{W}:\|W_i(t)\|_2 \leq k_i} \frac{1}{2}\|V(t) - (\mathbf{\Phi_W}\vec{x})(t)\|_2^2 \tag{13}$$

where $k_i$ is the current norm of $W_i(t)$. The constraints ensure that only the waveform shapes change (ideally, we would like the norm to be held fixed, but we relax to to an inequality to retain convexity), leaving any changes in scale to the previous step. Since $(\mathbf{\Phi_W}\vec{x})(t)$ is approximately a linear function of the waveforms, Eq. (13) is a standard ridge regression problem. Efficient algorithms exist for solving this problem in its dual form ([26]). This step is guaranteed to decrease the objective in Eq. (6) since the second term is held constant and the first term can only decrease.

## 4   Results

We applied our method to two data sets. The first was simulated according to the generative model described in Eq. (2-1). The second is real data from Harris et al. ([18]) consisting of simultaneous paired intracellular/extracellular recordings. The intracellular recording provides ground truth spikes for one of the cells in the extracellular recording.

### 4.1   Simulated data

We obtained three waveforms from retinal recordings made in the Chichilnisky lab at the Salk Institute (shown in Fig. 3(a)). Three Poisson spike trains were sampled independently with rate $(1-\rho)\lambda_0$ with $\lambda_0 = 10$Hz. To introduce a correlation of $\rho = \frac{1}{3}$, we sampled another Poisson spike train with rate $\rho\lambda_0$ and added these spikes (with random jitter) to each of the previous three trains. Spike amplitudes were drawn from $\mathcal{N}(1, 0.1^2)$. The spikes were convolved with the waveforms and Gaussian white noise was added (with $\sigma$ six times the smallest waveform amplitude). For clustering, the original trace was thresholded to identify segments(the threshold was varied in order to see the error tradeoff). PCA was applied and the leading PC's explaining 95% of the total variance were retained. $K$-means clustering was then applied (with $k = 3$) in the reduced space.

To reduce computational cost, we applied our method to disjoint segments of the trace, which were split off whenever activity was less than $3\sigma$ for more than half the waveform duration (about 4ms). The waveforms were initialized randomly and $P(\gamma)$ was Gamma-distributed with mean 0.0005 and coefficient of variation 0.25 (in Eq. (7)) for all experiments. The waveforms were allowed to change in length by adding (removing) padding on the ends on each iteration if the values exceeded (did not exceed) 5% of the peak amplitude (similar to [7]). Padding was added in increments of 10% of the current waveform length. Convex optimizations were performed using the CVX package ([27]). The learned waveforms and spike amplitude distributions are shown in Fig. 3. The amplitude distributions are well-matched to the generative distributions (shown in red). To evaluate performance, we counted missed spikes (relative to the number of true spikes) and false positives (relative to the number of predicted spikes) for clustering and our method. We varied the segment-finding threshold for clustering, and the amplitude threshold for our algorithm. The error tradeoff is shown in Fig. 4(a), and indicates that our method reduces both types of errors.

To visualize the errors, we chose optimal thresholds for each method (yielding the smallest number of misses and false positives), and then projected all segments used in clustering onto the first two principal components. We indicate by dots, open circles, and crosses the hits, misses, and false

positives, respectively (with colors indicating the waveform). For the same segments, we illustrate the behavior of our method in the same space. Note that unlike clustering, our method is allowed to assign more than one spike to each segment. The visualization is shown in Figures 4(b) and 4(c), and shows how clustering fails to account for the superimposed spikes, while our method eliminates a large portion of these errors. We found that this improvement was robust to the amount of noise added to the original trace (not shown).

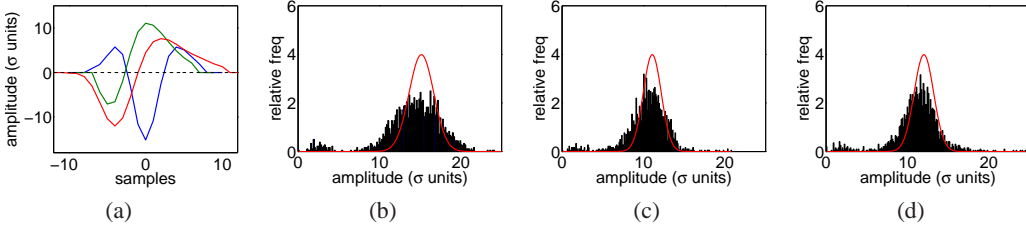

| (a) | (b) | (c) | (d) |

Figure 3: (a) Three waveforms used in simulations. (b),(c),(d) Histograms of the spike amplitudes learned by our algorithm of the blue,green, and red waveforms, respectively. The amplitudes were converted into units $\sigma$ by multiplying them by the corresponding waveform amplitudes, then dividing by the noise standard deviation. The red line indicates the generative density, corresponding to a Gaussian with mean 1 and standard deviation 0.1.

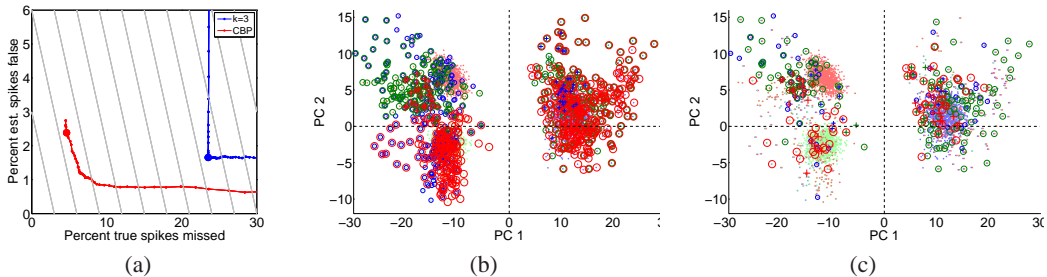

| (a) | (b) | (c) |

Figure 4: (a) Tradeoff of misses and false positives as the segment-identification threshold in clustering is varied (blue), and the amplitude threshold for our method (red) is varied. Diagonal lines indicate surfaces with equal total error. (b),(c) Visualization of spike sorting errors for clustering (b) and our method (c). Each point is a threshold-crossing segment in the signal, projected onto the first two principal components. Dots represent segments whose composite spikes were all correctly identified, with the color specifying the waveform (see Fig. 3(a)). Open circles and crosses represent misses and false positives, respectively. The thresholds were optimized for each method, and correspond to the enlarged dots in (a).

## 4.2 Real data

We used one electrode from the tetrode data in [18] to simplify our analysis. The raw trace was high-pass filtered (800Hz) to remove slow drift. The noise standard deviation was estimated from regions not exceeding three times the overall standard deviation. We then repeated the same analysis as for the simulated data. The resulting waveforms and coefficients histograms are shown in Figure 5. Unlike the simulated example, the spike amplitude distributions are bimodal in nature, despite the prior amplitude distribution containing only one Gaussian. We first focus on the high-amplitude groups (2 and 4), both of which are well-separated from their low-amplitude counterparts (1 and 3), suggesting that an appropriately chosen threshold would provide accurate spike identification for the ground-truth cell (4). Figure 6(a) confirms this, showing that our method provides substantial reduction in misses/false positives. Figures 6(b) and 6(c) show that, as before, the majority of this reduction is accounted for by recovering spikes overlapping with those of another cell (group 2). The low-amplitude groups (1 and 3) could arise from background cells whose waveforms look like scaled-down versions of those of the foreground cells 2 and 4, thus creating secondary "lumps" in the amplitude distributions. The projections of the events in these groups are labeled in Figures 6(b)

and 6(c), showing that it is unclear whether they arise from noise or one or two background cells. It is up to the user whether to interpret these badly-isolated groups as cells.

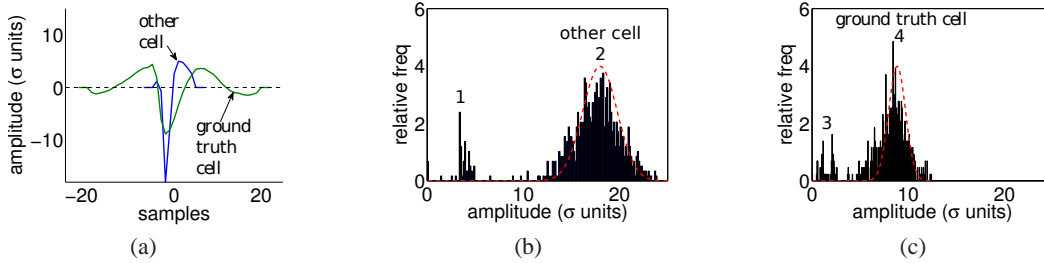

Figure 5: (a) Two waveforms learned from CBP. (b),(c) Distributions of the amplitude values for the blue and green waveform, respectively. The numbers label distinct groups of amplitudes that could be treated as spikes of a single cell. Group 4 corresponds to the ground truth cell. Group 2 corresponds to another foreground cell. Groups 1 and 3 likely correspond to a mixture of background cell activity and noise. The groups are labeled in PC-space in Figures 6(b) and 6(c).

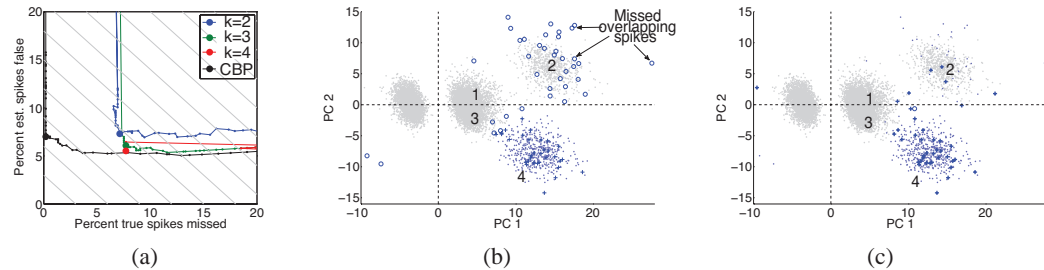

Figure 6: (a) Error tradeoff as in Fig. 4(a). The blue, green, and red curves are results of $k$-means clustering for different $k$. (b) Illustration of clustering errors in PC-space, with $k = 4$ and a threshold corresponding to the large red dot in (a). (c) Errors for our method with threshold corresponding to the large black dot. The numbers show the approximate location in PC-space of the amplitude groups demarcated in Figures 5(b) and 5(c).

## 5 Discussion

We have formulated the spike sorting problem as a maximum-a-posteriori (MAP) estimation problem, assuming a linear-Gaussian likelihood of the observed trace given the spikes and a Poisson process prior on the spikes. Unlike clustering methods, the model explicitly accounts for overlapping spikes, translation-invariance, and variability in spike amplitudes. Unlike other methods that handle overlapped spikes (e.g., [10]), our method jointly learns waveforms and spikes within a unified framework. We derived an iterative procedure based on block-coordinate descent to approximate the MAP solution. We showed empirically on simulated data that our method outperforms the standard clustering approach, particularly in the case of superimposed spikes. We also showed that our method yields an improvement on a real data set with ground truth, despite the fact that there are similar waveform shapes with different amplitudes. The majority of improvement in this case is also accounted for by identifying superimposed spikes. Our method has only a few parameters that are stable across a variety of conditions, thus addressing the need for an automated method for spike sorting that is not susceptible to systematic errors.

## Footnotes

[1] If $\mathbf{\Phi_W}$ is linear in $\mathbf{W}$, there is no change. For our choice of $\mathbf{\Phi_W}$, there is a small change of order $O(\Delta)$.

### References

[1] M. S. Lewicki. A review of methods for spike sorting: the detection and classification of neural action potentials. *Network*, 9(4):R53–R78, Nov 1998.

[2] M. Sahani. *Latent variable models for neural data analysis*. PhD thesis, California Institute of Technology, Pasadena, California, 1999.

[3] M Wehr, J S Pezaris, and M Sahani. Simultaneous paired intracellular and tetrode recordings for evaluating the performance of spike sorting algorithms. *Neurocomputing*, 26-27:1061–1068, 1999.

[4] Maneesh Sahani, John S. Pezaris, and Richard A. Andersen. On the separation of signals from neighboring cells in tetrode recordings. In *In Advances in Neural Information Processing Systems 10*, pages 222–228. MIT Press, 1998.

[5] P. H. van Cittert. Zum einflu der spaltbreite auf die intensittsverteilung in spektrallinien. ii. *Zeitschrift fr Physik A Hadrons and Nuclei*, 69:298–308, 1931. 10.1007/BF01391351.

[6] J. Mendel. *Optimal Seismic Deconvolution: An Estimation Based Approach*. Academic Press, 1983.

[7] Evan Smith and Michael S Lewicki. Efficient coding of time-relative structure using spikes. *Neural Computation*, 17(1):19–45, Jan 2005.

[8] Roger Grosse Rajat Raina, Helen Kwong, and Andrew Y. Ng. Shift-invariant sparse coding for audio classification. In *UAI*, 2007.

[9] J W Pillow, J Shlens, L Paninski, A Sher, A M Litke, E J Chichilnisky, and E P Simoncelli. Spatio-temporal correlations and visual signaling in a complete neuronal population. *Nature*, 454(7206):995–999, Aug 2008.

[10] Jason S. Prentice, Jan Homann, Kristina D. Simmons, Gaper Tkaik, Vijay Balasubramanian, and Philip C. Nelson. Fast, scalable, bayesian spike identification for multi-electrode arrays. *PLoS ONE*, 6(7):e19884, 07 2011.

[11] R. Quian Quiroga, Z. Nadasdy, and Y. Ben-Shaul. Unsupervised spike detection and sorting with wavelets and superparamagnetic clustering. *Neural Comput.*, 16:1661–1687, August 2004.

[12] Ki Yong Kwon and K. Oweiss. Wavelet footprints for detection and sorting of extracellular neural action potentials. In *Acoustics, Speech and Signal Processing (ICASSP), 2011 IEEE International Conference on*, pages 609 –612, may 2011.

[13] Markus Meister, Jerome Pine, and Denis A. Baylor. Multi-neuronal signals from the retina: acquisition and analysis. *Journal of Neuroscience Methods*, 51(1):95 – 106, 1994.

[14] S. Nirenberg, S. M. Carcieri, A. L. Jacobs, and P. E. Latham. Retinal ganglion cells act largely as independent encoders. *Nature*, 411:698–701, 2001.

[15] R. Segev, J. Goodhouse, J. Puchalla, and M. J. Berry. Recording spikes from a large fraction of the ganglion cells in a retinal patch. *Nature Neuroscience*, 7(10):1154–1161, October 2004.

[16] C. Pouzat, O. Mazor, and G. Laurent. Using noise signature to optimize spike-sorting and to assess neuronal classification quality. *J Neurosci Methods*, 122(1):43–57, 2002.

[17] Frank Wood, Michael J. Black, Carlos Vargas-irwin, Matthew Fellows, and John P. Donoghue. On the variability of manual spike sorting. *IEEE Transactions on Biomedical Engineering*, 51:912–918, 2004.

[18] Kenneth D. Harris, Darrell A. Henze, Jozsef Csicsvari, Hajime Hirase, Kenneth D, Darrell A. Henze, and Jozsef Csicsvari. Accuracy of tetrode spike separation as determined by simultaneous intracellular and extracellular measurements. *J Neurophysiol*, 84:401–414, 2000.

[19] Emery N. Brown, Robert E. Kass, and Partha P. Mitra. Multiple neural spike train data analysis: state-of-the-art and future challenges. *Nature neuroscience*, 7(5):456–461, May 2004.

[20] C Ekanadham, D Tranchina, and E P Simoncelli. Sparse decomposition of transformation-invariant signals with continuous basis pursuit. In *Proc. Int'l Conf Acoustics Speech Signal Processing (ICASSP)*, Los Angeles, CA, May 22-27 2011. IEEE Sig Proc Society.

[21] C Ekanadham, D Tranchina, and E P Simoncelli. Sparse decomposition of translation-invariant signals with continuous basis pursuit. *IEEE Transactions on Signal Processing*, 2011. Accepted for publication.

[22] B. A. Olshausen and D. J. Field. Emergence of simple-cell receptive field properties by learning a sparse code for natural images. *Nature*, 381(6583):607–609, Jun 1996.

[23] Ingrid Daubechies, Ronald DeVore, Massimo Fornasier, and C. Sinan Gntrk. Iteratively reweighted least squares minimization for sparse recovery. *Communications on Pure and Applied Mathematics*, 63(1):1–38, 2010.

[24] Emmanuel J. C. Enhancing sparsity by reweighted 1 minimization. *J. Fourier Analysis and Applications*, pages 877–905, 2008.

[25] R. Chartrand and Wotao Yin. Iteratively reweighted algorithms for compressive sensing. In *Acoustics, Speech and Signal Processing, 2008. ICASSP 2008. IEEE International Conference on*, pages 3869 –3872, 31 2008-april 4 2008.

[26] Honglak Lee, Alexis Battle, Rajat Raina, and Andrew Y. Ng. Efficient sparse coding algorithms. In *Advances in Neural Information Processing Systems 19*, pages 801–808. 2007.

[27] M. Grant and S. Boyd. CVX: Matlab software for disciplined convex programming, version 1.21. http://cvxr.com/cvx, October 2010.

